# Complexity of Inference in Latent Dirichlet Allocation

**David Sontag**
New York University*

**Daniel M. Roy**
University of Cambridge

## Abstract

We consider the computational complexity of probabilistic inference in Latent Dirichlet Allocation (LDA). First, we study the problem of finding the maximum a posteriori (MAP) assignment of topics to words, where the document's topic distribution is integrated out. We show that, when the effective number of topics per document is small, exact inference takes polynomial time. In contrast, we show that, when a document has a large number of topics, finding the MAP assignment of topics to words in LDA is NP-hard. Next, we consider the problem of finding the MAP topic distribution for a document, where the topic-word assignments are integrated out. We show that this problem is also NP-hard. Finally, we briefly discuss the problem of sampling from the posterior, showing that this is NP-hard in one restricted setting, but leaving open the general question.

## 1 Introduction

Probabilistic models of text and topics, known as topic models, are powerful tools for exploring large data sets and for making inferences about the content of documents. Topic models are frequently used for deriving low-dimensional representations of documents that are then used for information retrieval, document summarization, and classification [Blei and McAuliffe, 2008; Lacoste-Julien et al., 2009]. In this paper, we consider the computational complexity of inference in topic models, beginning with one of the simplest and most popular models, Latent Dirichlet Allocation (LDA) [Blei et al., 2003]. The LDA model is arguably one of the most important probabilistic models in widespread use today.

Almost all uses of topic models require probabilistic inference. For example, unsupervised learning of topic models using Expectation Maximization requires the repeated computation of marginal probabilities of what topics are present in the documents. For applications in information retrieval and classification, each new document necessitates inference to determine what topics are present.

Although there is a wealth of literature on approximate inference algorithms for topic models, such Gibbs sampling and variational inference [Blei et al., 2003; Griffiths and Steyvers, 2004; Mukherjee and Blei, 2009; Porteous et al., 2008; Teh et al., 2007], little is known about the computational complexity of exact inference. Furthermore, the existing inference algorithms, although well-motivated, do not provide guarantees of optimality. We choose to study LDA because we believe that it captures the essence of what makes inference easy or hard in topic models. We believe that a careful analysis of the complexity of popular probabilistic models like LDA will ultimately help us build a methodology for spanning the gap between theory and practice in probabilistic AI.

Our hope is that our results will motivate discussion of the following questions, guiding research of both new topic models and the design of new approximate inference and learning

algorithms. First, what is the structure of real-world LDA inference problems? Might there be structure in "natural" problem instances that makes them different from hard instances (e.g., those used in our reductions)? Second, how strongly does the prior distribution bias the results of inference? How do the hyperparameters affect the structure of the posterior and the hardness of inference?

We study the complexity of finding assignments of topics to words with high posterior probability and the complexity of summarizing the posterior distributions on topics in a document by either its expectation or points with high posterior density. In the former case, we show that the number of topics in the maximum a posteriori assignment determines the hardness. In the latter case, we quantify the sense in which the Dirichlet prior can be seen to enforce sparsity and use this result to show hardness via a reduction from set cover.

## 2  MAP inference of word assignments

We will consider the inference problem for a single document. The LDA model states that the document, represented as a collection of words $\mathbf{w} = (w_1, w_2, \ldots, w_N)$, is generated as follows: a distribution over the $T$ topics is sampled from a Dirichlet distribution, $\theta \sim \mathrm{Dir}(\alpha)$; then, for $i \in [N] := \{1, \ldots, N\}$, we sample a topic $z_i \sim \mathrm{Multinomial}(\theta)$ and word $w_i \sim \Psi_{z_i}$, where $\Psi_t$, $t \in [T]$ are distributions on a dictionary of words. Assume that the word distributions $\Psi_t$ are fixed (e.g., they have been previously estimated), and let $l_{it} = \log \Pr(w_i | z_i = t)$ be the log probability of the $i$th word being generated from topic $t$. After integrating out the topic distribution vector, the joint distribution of the topic assignments conditioned on the words $\mathbf{w}$ is given by

$$\Pr(z_1, \ldots, z_N | \mathbf{w}) \propto \frac{\Gamma(\sum_t \alpha_t)}{\prod_t \Gamma(\alpha_t)} \frac{\prod_t \Gamma(n_t + \alpha_t)}{\Gamma(\sum_t \alpha_t + N)} \prod_{i=1}^{N} \Pr(w_i | z_i), \tag{1}$$

where $n_t$ is the total number of words assigned to topic $t$.

In this section, we focus on the inference problem of finding the most likely assignment of topics to words, i.e. the maximum a posteriori (MAP) assignment. This has many possible applications. For example, it can be used to cluster the words of a document, or as part of a larger system such as part-of-speech tagging [Li and McCallum, 2005]. More broadly, for many classification tasks involving topic models it may be useful to have word-level features for whether a particular word was assigned to a given topic. From both an algorithm design and complexity analysis point of view, this MAP problem has the additional advantage of involving only discrete random variables.

Taking the logarithm of Eq. 1 and ignoring constants, finding the MAP assignment is seen to be equivalent to the following combinatorial optimization problem:

$$\Phi = \max_{x_{it} \in \{0,1\}, n_t} \quad \sum_t \log \Gamma(n_t + \alpha_t) + \sum_{i,t} x_{it} l_{it} \tag{2}$$

$$\text{subject to} \quad \sum_t x_{it} = 1, \quad \sum_i x_{it} = n_t,$$

where the indicator variable $x_{it} = \mathbb{I}[z_i = t]$ denotes the assignment of word $i$ to topic $t$.

### 2.1  Exact maximization for small number of topics

Suppose a document only uses $\tau \ll T$ topics. That is, $T$ could be large, but we are guaranteed that the MAP assignment for a document uses at most $\tau$ different topics. In this section, we show how we can use this knowledge to efficiently find a maximizing assignment of words to topics. It is important to note that we only restrict the *maximum* number of topics per document, letting the Dirichlet prior and the likelihood guide the choice of the actual number of topics present.

We first observe that, if we knew the *number* of words assigned to each topic, finding the MAP assignment is easy. For $t \in [T]$, let $n_t^*$ be the number of words assigned to topic $t$

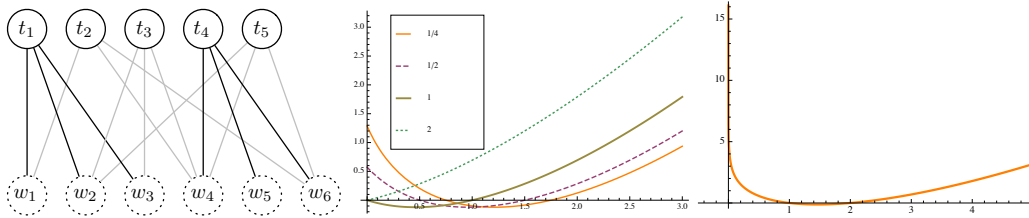

**Figure 1:** **(Left)** A LDA instance derived from a $k$-set packing instance. **(Center)** Plot of $F(n_t) = \log\Gamma(n_t + \alpha)$ for various values of $\alpha$. The $x$-axis varies $n_t$, the number of words assigned to topic $t$, and the $y$-axis shows $F(n_t)$. **(Right)** Behavior of $\log\Gamma(n_t + \alpha)$ as $\alpha \to 0$. The function is stable everywhere but at zero, where the reward for sparsity increases without bound.

in the MAP assignment. Then, the MAP assignment $\mathbf{x}$ is found by solving the following optimization problem:

$$\max_{x_{it} \in \{0,1\}} \quad \sum_{i,t} x_{it} l_{it} \tag{3}$$
$$\text{subject to} \quad \sum_{t} x_{it} = 1, \quad \sum_{i} x_{it} = n_t^*,$$

which is equivalent to weighted $b$-matching in a bipartite graph (the words are on one side, the topics on the other) and can be optimally solved in time $O(bm^3)$, where $b = \max_t n_t^* = O(N)$ and $m = N + T$ [Schrijver, 2003].

We call $(n_1, \ldots, n_T)$ a *valid partition* when $n_i \geq 0$ and $\sum_t n_t = N$. Using weighted $b$-matching, we can find a MAP assignment of words to topics by trying all $\binom{T}{\tau} = \Theta(T^\tau)$ choices of $\tau$ topics and all possible valid partitions with at most $\tau$ non-zeros.

---

> **for all** subsets $A \subseteq [T]$ such that $|A| = \tau$ **do**
>     **for all** valid partitions $\mathbf{n} = (n_1, n_2, \ldots, n_T)$ such that $n_t = 0$ for $t \notin A$ **do**
>        $\Phi_{A,\mathbf{n}} \leftarrow \text{Weighted-B-Matching}(A, \mathbf{n}, l) + \sum_t \log\Gamma(n_t + \alpha_t)$
>     **end for**
> **end for**
> **return** $\arg\max_{A,\mathbf{n}} \Phi_{A,\mathbf{n}}$

---

There are at most $N^{\tau-1}$ valid partitions with $\tau$ non-zero counts. For each of these, we solve the $b$-matching problem to find the most likely assignment of words to topics that satisfies the cardinality constraints. Thus, the total running time is $O((NT)^\tau (N + \tau)^3)$. This is polynomial when the number of topics $\tau$ appearing in a document is a constant.

## 2.2 Inference is NP-hard for large numbers of topics

In this section, we show that probabilistic inference is NP-hard in the general setting where a document may have a large number of topics in its MAP assignment. Let WORD-LDA($\alpha$) denote the decision problem of whether $\Phi > V$ (see Eq. 2) for some $V \in \mathbb{R}$, where the hyperparameters $\alpha_t = \alpha$ for all topics. We consider both $\alpha < 1$ and $\alpha \geq 1$ because, as shown in Figure 1, the optimization problem is qualitatively different in these two cases.

**Theorem 1.** WORD-LDA($\alpha$) *is NP-hard for all* $\alpha > 0$.

*Proof.* Our proof is a straightforward generalization of the approach used by Halperin and Karp [2005] to show that the minimum entropy set cover problem is hard to approximate.

The proof is done by reduction from $k$-set packing ($k$-SP), for $k \geq 3$. In $k$-SP, we are given a collection of $k$-element sets over some universe of elements $\Sigma$ with $|\Sigma| = n$. The goal is to find the largest collection of *disjoint* sets. There exists a constant $c < 1$ such that it is NP-hard to decide whether a $k$-SP instance has (i) a solution with $n/k$ disjoint sets

covering all elements (called a *perfect matching*), or (ii) at most $cn/k$ disjoint sets (called a $(cn/k)$-matching).

We now describe how to construct a LDA inference problem from a $k$-SP instance. This requires specifying the words in the document, the number of topics, and the word log probabilities $l_{it}$. Let each element $i \in \Sigma$ correspond to a word $w_i$, and let each set correspond to one topic. The document consists of all of the words (i.e., $\Sigma$). We assign uniform probability to the words in each topic, so that $\Pr(w_i|z_i = t) = \frac{1}{k}$ for $i \in t$, and 0 otherwise. Figure 1 illustrates the resulting LDA model. The topics are on the top, and the words from the document are on the bottom. An edge is drawn between a topic (set) and a word (element) if the corresponding set contains that element.

What remains is to show that we can solve some $k$-SP problem by using this reduction and solving a WORD-LDA($\alpha$) problem. For technical reasons involving $\alpha > 1$, we require that $k$ is sufficiently large. We will use the following result (we omit the proof due to space limitations).

**Lemma 2.** *Let $P$ be a $k$-SP instance for $k > (1 + \alpha)^2$, and let $P'$ be the derived WORD-LDA($\alpha$) instance. There exist constants $C_U$ and $C_L < C_U$ such that, if there is a perfect matching in $P$, then $\Phi \geq C_U$. If, on the other hand, there is at most a $(cn/k)$-matching in $P$, then $\Phi < C_L$.*

Let $P$ be a $k$-SP instance for $k > (3 + \alpha)^2$, $P'$ be the derived WORD-LDA($\alpha$) instance, and $C_U$ and $C_L < C_U$ be as in Lemma 2. Then, by testing $\Phi < C_L$ and $\Phi > C_U$ we can decide whether $P$ is a perfect matching or at best a $(cn/k)$-matching. Hence $k$-SP reduces to WORD-LDA($\alpha$). □

The bold lines in Figure 1 indicate the MAP assignment, which for this example corresponds to a perfect matching for the original $k$-set packing instance. More realistic documents would have significantly more words than topics used. Although this is not possible while keeping $k = 3$, since the MAP assignment always has $\tau \geq N/k$, we can instead reduce from a $k$-set packing problem with $k \gg 3$. Lemma 2 shows that this is hard as well.

## 3 MAP inference of the topic distribution

In this section we consider the task of finding the mode of $\Pr(\theta|\mathbf{w})$. This MAP problem involves integrating out the topic assignments, $z_i$, as opposed to the previously considered MAP problem of integrating out the topic distribution $\theta$. We will see that the MAP topic distribution is not always well-defined, which will lead us to define and study alternative formulations. In particular, we give a precise characterization of the MAP problem as one of finding sparse topic distributions, and use this fact to give hardness results for several settings. We also show settings for which MAP inference is tractable.

There are many potential applications of MAP inference of the document's topic distribution. For example, the distribution may be used for topic-based information retrieval or as the feature vector for classification. As we will make clear later, this type of inference results in *sparse* solutions. Thus, the MAP topic distribution provides a compact summary of the document that could be useful for document summarization.

Let $\theta = (\theta_1, \dots, \theta_T)$. A straightforward application of Bayes' rule allows us to write the posterior density of $\theta$ given $\mathbf{w}$ as

$$\Pr(\theta|\mathbf{w}) \propto \left( \prod_{t=1}^{T} \theta_t^{\alpha_t - 1} \right) \left( \prod_{i=1}^{N} \sum_{t=1}^{T} \theta_t \psi_{it} \right), \tag{4}$$

where $\psi_{it} = \Pr(w_i|z_i = t)$. Taking the logarithm of the posterior and ignoring constants, we obtain

$$\Phi(\theta) = \sum_{t=1}^{T} (\alpha_t - 1) \log(\theta_t) + \sum_{i=1}^{N} \log \left( \sum_{t=1}^{T} \theta_t \psi_{it} \right) \tag{5}$$

We will use the shorthand $\Phi(\theta) = P(\theta) + L(\theta)$, where $P(\theta) = \sum_{t=1}^{T}(\alpha_t - 1)\log(\theta_t)$ and $L(\theta) = \sum_{i=1}^{N}\log(\sum_{t=1}^{T}\psi_{it}\theta_t)$.

To find the MAP $\theta$, we maximize (5) subject to the constraint that $\sum_{t=1}^{T}\theta_t = 1$ and $\theta_t \geq 0$. Unfortunately, this maximization problem can be degenerate. In particular, note that if $\theta_t = 0$ for $\alpha_t < 1$, then the corresponding term in $P(\theta)$ will take the value $\infty$, overwhelming the likelihood term. Thus, any feasible solution with the above property could be considered 'optimal'.

A similar problem arises during the maximum-likelihood estimation of a normal mixture model, where the likelihood diverges to infinity as the variance of a mixture component with a single data point approaches zero [Biernacki and Chrétien, 2003; Kiefer and Wolfowitz, 1956]. In practice, one can enforce a lower bound on the variance or penalize such configurations. Here we consider a similar tactic.

For $\epsilon > 0$, let TOPIC-LDA($\epsilon$) denote the optimization problem

$$\max_{\theta} \ \Phi(\theta) \quad \text{subject to} \quad \sum_{t} \theta_t = 1, \quad \epsilon \leq \theta_t \leq 1. \tag{6}$$

For $\epsilon = 0$, we will denote the corresponding optimization problem by TOPIC-LDA. When $\alpha_t = \alpha$, i.e. the prior distribution on the topic distribution is a symmetric Dirichlet, we write TOPIC-LDA($\epsilon,\alpha$) for the corresponding optimization problem. In the following sections we will study the structure and hardness of TOPIC-LDA, TOPIC-LDA($\epsilon$) and TOPIC-LDA($\epsilon,\alpha$).

## 3.1 Polynomial-time inference for large hyperparameters $(\alpha_t \geq 1)$

When $\alpha_t \geq 1$, Eq. 5 is a concave function of $\theta$. As a result, we can efficiently find $\theta^*$ using a number of techniques from convex optimization. Note that this is in contrast to the MAP inference problem discussed in Section 2, which we showed was hard for all choices of $\alpha$.

Since we are optimizing over the simplex ($\theta$ must be non-negative and sum to 1), we can apply the exponentiated gradient method [Kivinen and Warmuth, 1995]. Initializing $\theta^0$ to be the uniform vector, the update for time $s$ is given by

$$\theta_t^{s+1} = \frac{\theta_t^s \exp(\eta \nabla_t^s)}{\sum_{\hat{t}} \theta_{\hat{t}}^s \exp(\eta \nabla_{\hat{t}}^s)}, \qquad \nabla_t^s = \frac{\alpha_t - 1}{\theta_t^s} + \sum_{i=1}^{N} \frac{\psi_{it}}{\sum_{\hat{t}=1}^{T} \theta_{\hat{t}}^s \psi_{i\hat{t}}}, \tag{7}$$

where $\eta$ is the step size and $\nabla^s$ is the gradient.

When $\alpha = 1$ the prior disappears altogether and this algorithm simply corresponds to optimizing the likelihood term. When $\alpha \gg 1$, the prior corresponds to a bias toward a particular $\theta$ topic distribution.

## 3.2 Small hyperparameters encourage sparsity $(\alpha < 1)$

On the other hand, when $\alpha_t < 1$, the first term in Eq. 5 is convex whereas the second term is concave. This setting, of $\alpha$ much smaller than 1, occurs frequently in practice. For example, learning a LDA model on a large corpus of NIPS abstracts with $T = 200$ topics, we find that the hyperparameters found range from $\alpha_t = 0.0009$ to $0.135$, with the median being $0.01$. Although in this setting it is difficult to find the global optimum (we will make this precise in Theorem 6), one possibility for finding a local maximum is the Concave-Convex Procedure [Yuille and Rangarajan, 2003].

In this section we prove structural results about the TOPIC-LDA($\epsilon,\alpha$) solution space for when $\alpha < 1$. These results illustrate that the Dirichlet prior encourages *sparse* MAP solutions: the topic distribution will be large on as few topics as necessary to explain every word of the document, and otherwise will be close to zero.

The following lemma shows that in any optimal solution to TOPIC-LDA($\epsilon,\alpha$), for every word, there is at least one topic that both has large probability and gives non-trivial probability to this word. We use $K(\alpha, T, N) = e^{-3/\alpha} N^{-1} T^{-1/\alpha}$ to refer to the lower bound on the topic's probability.

**Lemma 3.** *Let $\alpha < 1$. All optimal solutions $\theta^*$ to TOPIC-LDA($\epsilon,\alpha$) have the following property: for every word $i$, $\theta_{\hat{t}}^* \geq K(\alpha, T, N)$ where $\hat{t} = \arg\max_t \psi_{it}\theta_t^*$.*

*Proof sketch.* If $\epsilon \geq K(\alpha, T, N)$ the claim trivially holds. Assume for the purpose of contradiction that there exists a word $\hat{i}$ such that $\theta_{\hat{t}}^* < K(\alpha, T, N)$, where $\hat{t} = \arg\max_t \psi_{\hat{i}t}\theta_t^*$.

Let $Y$ denote the set of topics $t \neq \hat{t}$ such that $\theta_t^* \geq 2\epsilon$. Let $\beta_1 = \sum_{t \in Y} \theta_t^*$ and $\beta_2 = \sum_{t \notin Y, t \neq \hat{t}} \theta_t^*$. Note that $\beta_2 < 2T\epsilon$. Consider

$$\hat{\theta}_{\hat{t}} = \frac{1}{n}, \qquad \hat{\theta}_t = \left(\frac{1 - \beta_2 - \frac{1}{n}}{\beta_1}\right)\theta_t^* \text{ for } t \in Y, \qquad \hat{\theta}_t = \theta_t^* \text{ for } t \notin Y, t \neq \hat{t}. \tag{8}$$

It is easy to show that $\forall t$, $\hat{\theta}_t \geq \epsilon$, and $\sum_t \hat{\theta}_t = 1$. Finally, we show that $\Phi(\hat{\theta}) > \Phi(\theta^*)$, contradicting the optimality of $\theta^*$. The full proof is given in the supplementary material. $\square$

Next, we show that if a topic is not sufficiently "used" then it will be given a probability very close to zero. By used, we mean that for at least one word, the topic is close in probability to that of the largest contributor to the likelihood of the word. To do this, we need to define the notion of the *dynamic range* of a word, given as $\kappa_i = \max_{t,t':\psi_{it}>0,\psi_{it'}>0} \frac{\psi_{it}}{\psi_{it'}}$. We let the maximum dynamic range be $\kappa = \max_i \kappa_i$. Note that $\kappa \geq 1$ and, for most applications, it is reasonable to expect $\kappa$ to be small (e.g., less than 1000).

**Lemma 4.** *Let $\alpha < 1$, and let $\theta^*$ be any optimal solution to TOPIC-LDA($\epsilon,\alpha$). Suppose topic $\hat{t}$ has $\theta_{\hat{t}}^* < (\kappa N)^{-1} K(\alpha, T, N)$. Then, $\theta_{\hat{t}}^* \leq e^{\frac{1}{1-\alpha}+2}\epsilon$.*

*Proof.* Suppose for the purpose of contradiction that $\theta_{\hat{t}}^* > e^{\frac{1}{1-\alpha}+2}\epsilon$. Consider $\hat{\theta}$ defined as follows: $\hat{\theta}_{\hat{t}} = \epsilon$, and $\hat{\theta}_t = \left(\frac{1-\epsilon}{1-\theta_{\hat{t}}^*}\right)\theta_t^*$ for $t \neq \hat{t}$. We have:

$$\Phi(\hat{\theta}) - \Phi(\theta^*) = (1 - \alpha)\log\left(\frac{\theta_{\hat{t}}^*}{\epsilon}\right) + (T-1)(1-\alpha)\log\left(\frac{1 - \theta_{\hat{t}}^*}{1-\epsilon}\right) + \sum_{i=1}^{N} \log\left(\frac{\sum_t \hat{\theta}_t \psi_{it}}{\sum_t \theta_t^* \psi_{it}}\right).$$

Using the fact that $\log(1 - z) \geq -2z$ for $z \in [0, \frac{1}{2}]$, it follows that

$$(T-1)(1-\alpha)\log\left(\frac{1-\theta_{\hat{t}}^*}{1-\epsilon}\right) \geq (T-1)(1-\alpha)\log\left(1 - \theta_{\hat{t}}^*\right) \geq 2(T-1)(\alpha - 1)\theta_{\hat{t}}^* \tag{9}$$

$$\geq 2(T-1)(\alpha - 1)(\kappa N)^{-1}K(\alpha, T, N) \geq 2(\alpha - 1). \tag{10}$$

We have $\hat{\theta}_t \geq \theta_t^*$ for $t \neq \hat{t}$, and so

$$\frac{\sum_t \hat{\theta}_t \psi_{it}}{\sum_t \theta_t^* \psi_{it}} = \frac{\sum_{t \neq \hat{t}} \hat{\theta}_t \psi_{it} + \epsilon\psi_{i\hat{t}}}{\sum_{t \neq \hat{t}} \theta_t^* \psi_{it} + \theta_{\hat{t}}^* \psi_{i\hat{t}}} \geq \frac{\sum_{t \neq \hat{t}} \theta_t^* \psi_{it}}{\sum_{t \neq \hat{t}} \theta_t^* \psi_{it} + \theta_{\hat{t}}^* \psi_{i\hat{t}}}. \tag{11}$$

Recall from Lemma 3 that, for each word $i$ and $\tilde{t} = \arg\max_t \psi_{it}\theta_t^*$, we have $\theta_{\tilde{t}} > K(\alpha, T, N)$. Necessarily $\tilde{t} \neq \hat{t}$. Therefore, using the fact that $\log\frac{1}{1+z} \geq -z$,

$$\log\left(\frac{\sum_{t \neq \hat{t}} \theta_t^* \psi_{it}}{\sum_{t \neq \hat{t}} \theta_t^* \psi_{it} + \theta_{\hat{t}}^* \psi_{i\hat{t}}}\right) \geq -\frac{\theta_{\hat{t}}^* \psi_{i\hat{t}}}{\sum_{t \neq \hat{t}} \theta_t^* \psi_{it}} \geq -\frac{(\kappa N)^{-1}K(\alpha, T, N)\psi_{i\hat{t}}}{K(\alpha, T, N)\psi_{i\tilde{t}}} \geq -\frac{1}{n}. \tag{12}$$

Thus, $\Phi(\hat{\theta}) - \Phi(\theta^*) > (1-\alpha)\log e^{\frac{1}{1-\alpha}+2} + 2(\alpha - 1) - 1 = 0$, completing the proof. $\square$

Finally, putting together what we showed in the previous two lemmas, we conclude that all optimal solutions to TOPIC-LDA($\epsilon,\alpha$) either have $\theta_t$ large or small, but not in between (that is, we have demonstrated a gap). We have the immediate corollary:

**Theorem 5.** *For $\alpha < 1$, all optimal solutions to TOPIC-LDA($\epsilon,\alpha$) have $\theta_t \leq \left(e^{\frac{1}{1-\alpha}+2}\right)\epsilon$ or $\theta_t \geq \kappa^{-1}e^{-3/\alpha}N^{-2}T^{-1/\alpha}$.*

### 3.3 Inference is NP-hard for small hyperparameters ($\alpha < 1$)

The previous results characterize optimal solutions to TOPIC-LDA($\epsilon,\alpha$) and highlight the fact that optimal solutions are sparse. In this section we show that these same properties can be the source of computational hardness during inference. In particular, it is possible to encode set cover instances as TOPIC-LDA($\epsilon,\alpha$) instances, where the set cover corresponds to those topics assigned appreciable probability.

**Theorem 6.** *TOPIC-LDA($\epsilon,\alpha$) is NP-hard for $\epsilon \leq K(\alpha, T, N)^{T/(1-\alpha)} T^{N/(1-\alpha)}$ and $\alpha < 1$.*

*Proof.* Consider a set cover instance consisting of a universe of elements and a family of sets, where we assume for convenience that the minimum cover is neither a singleton, all but one of the family of sets, nor the entire family of sets, and that there are at least two elements in the universe. As with our previous reduction, we have one topic per set and one word in the document for each element. We let $\Pr(w_i|z_i = t) = 0$ when element $w_i$ is not in set $t$, and a constant otherwise (we make every topic have the uniform distribution over the same number of words, some of which may be dummy words not appearing in the document). Let $S_i \subseteq [T]$ denote the set of topics to which word $i$ belongs. Then, up to additive constants, we have $P(\theta) = -(1 - \alpha) \sum_{t=1}^{T} \log(\theta_t)$ and $L(\theta) = \sum_{i=1}^{N} \log(\sum_{t \in S_i} \theta_t)$.

Let $C_{\theta^*} \subseteq [T]$ be those topics $t \in [T]$ such that $\theta_t^* \geq K(\alpha, T, n)$, where $\theta^*$ is an optimal solution to TOPIC-LDA($\epsilon,\alpha$). It immediately follows from Lemma 3 that $C_{\theta^*}$ is a cover.

Suppose for the purpose of contradiction that $C_{\theta^*}$ is not a minimal cover. Let $\tilde{C}$ be a minimal cover, and let $\tilde{\theta}_t = \epsilon$ for $t \notin \tilde{C}$ and $\tilde{\theta}_t = \frac{1 - \epsilon(T - |\tilde{C}|)}{|\tilde{C}|} > \frac{1}{T}$ otherwise. We will show that $\Phi(\tilde{\theta}) > \Phi(\theta^*)$, contradicting the optimality of $\theta^*$, and thus proving that $C_{\theta^*}$ is in fact minimal. This suffices to show that TOPIC-LDA($\epsilon,\alpha$) is NP-hard in this regime.

For all $\theta$ in the simplex, we have $\sum_i \log(\max_{t \in S_i} \theta_t) \leq L(\theta) \leq 0$. Thus it follows that $L(\theta^*) - L(\tilde{\theta}) \leq N \log T$. Likewise, using the assumption that $T \geq |\tilde{C}| + 1$, we have

$$\frac{P(\tilde{\theta}) - P(\theta^*)}{(1 - \alpha)} \geq -(T - |\tilde{C}|) \log \epsilon - (|\tilde{C}| + 1) \log K(\alpha, T, N) + (T - |\tilde{C}| - 1) \log \epsilon \quad (13)$$

$$\geq \log \frac{1}{\epsilon} - T \log K(\alpha, T, N), \quad (14)$$

where we have conservatively only included the terms $t \notin \tilde{C}$ for $P(\tilde{\theta})$ and taken $\theta^* \in \{\epsilon, K(\alpha, T, N)\}$ with $|\tilde{C}| + 1$ terms taking the latter value. It follows that

$$\left(P(\tilde{\theta}) + L(\tilde{\theta})\right) - \left(P(\theta^*) + L(\theta^*)\right) > (1 - \alpha) \log \frac{1}{\epsilon} - (T \log K(\alpha, T, N) + N \log T). \quad (15)$$

This is greater than 0 precisely when $(1 - \alpha) \log \frac{1}{\epsilon} > \log T^N K(\alpha, T, N)^T$. $\qquad\square$

Note that although $\epsilon$ is exponentially small in $N$ and $T$, the size of its representation in binary is polynomial in $N$ and $T$, and thus polynomial in the size of the set cover instance.

It can be shown that as $\epsilon \to 0$, the solutions to TOPIC-LDA($\epsilon,\alpha$) become degenerate, concentrating their support on the minimal set of topics $C \subseteq [T]$ such that $\forall i, \exists t \in C$ s.t. $\psi_{it} > 0$. A generalization of this result holds for TOPIC-LDA($\epsilon$) and suggests that, while it may be possible to give a more sensible definition of TOPIC-LDA as the set of solutions for TOPIC-LDA($\epsilon$) as $\epsilon \to 0$, these solutions are unlikely to be of any practical use.

## 4 Sampling from the posterior

The previous sections of the paper focused on MAP inference problems. In this section, we study the problem of marginal inference in LDA.

**Theorem 7.** *For $\alpha > 1$, one can approximately sample from $\Pr(\theta \mid \mathbf{w})$ in polynomial time.*

*Proof sketch.* The density given in Eq. 4 is log-concave when $\alpha \geq 1$. The algorithm given in Lovasz and Vempala [2006] can be used to approximately sample from the posterior. $\quad\square$

Although polynomial, it is not clear whether the algorithm given in Lovasz and Vempala [2006], based on random walks, is of practical interest (e.g., the running time bound has a constant of $10^{30}$). However, we believe our observation provides insight into the complexity of sampling when $\alpha$ is not too small, and may be a starting point towards explaining the empirical success of using Markov chain Monte Carlo to do inference in LDA.

Next, we show that when $\alpha$ is *extremely* small, it is NP-hard to sample from the posterior. We again reduce from set cover. The intuition behind the proof is that, when $\alpha$ is small enough, an appreciable amount of the probability mass corresponds to the sparsest possible $\theta$ vectors where the supported topics together cover all of the words. As a result, we could directly read off the minimal set cover from the posterior marginals $\mathbb{E}[\theta_t \mid \mathbf{w}]$.

**Theorem 8.** *When $\alpha < \left((4N + 4)T^N \Gamma(N)\right)^{-1}$, it is NP-hard to approximately sample from* $\Pr(\theta \mid \mathbf{w})$*, under randomized reductions.*

The full proof can be found in the supplementary material. Note that it is likely that one would need an extremely large and unusual corpus to learn an $\alpha$ so small. Our results illustrate a large gap in our knowledge about the complexity of sampling as a function of $\alpha$. We feel that tightening this gap is a particularly exciting open problem.

## 5  Discussion

In this paper, we have shown that the complexity of MAP inference in LDA strongly depends on the effective number of topics per document. When a document is generated from a small number of topics (regardless of the number of topics in the model), WORD-LDA can be solved in polynomial time. We believe this is representative of many real-world applications. On the other hand, if a document can use an arbitrary number of topics, WORD-LDA is NP-hard. The choice of hyperparameters for the Dirichlet does not affect these results.

We have also studied the problem of computing MAP estimates and expectations of the topic distribution. In the former case, the Dirichlet prior enforces sparsity in a sense that we make precise. In the latter case, we show that extreme parameterizations can similarly cause the posterior to concentrate on sparse solutions. In both cases, this sparsity is shown to be a source of computational hardness.

In related work, Seppänen et al. [2003] suggest a heuristic for inference that is also applicable to LDA: if there exists a word that can only be generated with high probability from one of the topics, then the corresponding topic must appear in the MAP assignment whenever that word appears in a document. Miettinen et al. [2008] give a hardness reduction and greedy algorithm for learning topic models. Although the models they consider are very different from LDA, some of the ideas may still be applicable. More broadly, it would be interesting to consider the complexity of learning the per-topic word distributions $\Psi_t$.

Our paper suggests a number of directions for future study. First, our exact algorithms can be used to evaluate the accuracy of approximate inference algorithms, for example by comparing to the MAP of the variational posterior. On the algorithmic side, it would be interesting to improve the running time of the exact algorithm from Section 2.1. Also, note that we did not give an analogous exact algorithm for the MAP topic distribution when the posterior has support on only a small number of topics. In this setting, it may be possible to find this set of topics by trying all $S \subseteq [T]$ of small cardinality and then doing a (non-uniform) grid search over the topic distribution restricted to support $S$.

Finally, our structural results on the sparsity induced by the Dirichlet prior draws connections between inference in topic models and sparse signal recovery. We proved that the MAP topic distribution has, for each topic $t$, either $\theta_t \approx \epsilon$ or $\theta_t$ bounded below by some value (much larger than $\epsilon$). Because of this gap, we can approximately view the MAP problem as searching for a set corresponding to the support of $\theta$. Our work motivates the study of greedy algorithms for MAP inference in topic models, analogous to those used for set cover. One could even consider learning algorithms that use this greedy algorithm within the inner loop [Krause and Cevher, 2010].

**Acknowledgments**  D.M.R. is supported by a Newton International Fellowship.  We thank Tommi Jaakkola and anonymous reviewers for helpful comments.

## Footnotes

* This work was partially carried out while D.S. was at Microsoft Research New England.

# References

C. Biernacki and S. Chrétien. Degeneracy in the maximum likelihood estimation of univariate Gaussian mixtures with EM. *Statist. Probab. Lett.*, 61(4):373–382, 2003. ISSN 0167-7152.

D. Blei and J. McAuliffe. Supervised topic models. In J. Platt, D. Koller, Y. Singer, and S. Roweis, editors, *Adv. in Neural Inform. Processing Syst. 20*, pages 121–128. MIT Press, Cambridge, MA, 2008.

D. M. Blei, A. Y. Ng, and M. I. Jordan. Latent Dirichlet allocation. *J. Mach. Learn. Res.*, 3: 993–1022, 2003. ISSN 1532-4435.

T. L. Griffiths and M. Steyvers. Finding scientific topics. *Proc. Natl. Acad. Sci. USA*, 101(Suppl 1):5228–5235, 2004. doi: 10.1073/pnas.0307752101.

E. Halperin and R. M. Karp. The minimum-entropy set cover problem. *Theor. Comput. Sci.*, 348 (2):240–250, 2005. ISSN 0304-3975. doi: http://dx.doi.org/10.1016/j.tcs.2005.09.015.

J. Kiefer and J. Wolfowitz. Consistency of the maximum likelihood estimator in the presence of infinitely many incidental parameters. *Ann. Math. Statist.*, 27:887–906, 1956. ISSN 0003-4851.

J. Kivinen and M. K. Warmuth. Exponentiated gradient versus gradient descent for linear predictors. *Inform. and Comput.*, 132, 1995.

A. Krause and V. Cevher. Submodular dictionary selection for sparse representation. In *Proc. Int. Conf. on Machine Learning (ICML)*, 2010.

S. Lacoste-Julien, F. Sha, and M. Jordan. DiscLDA: Discriminative learning for dimensionality reduction and classification. In D. Koller, D. Schuurmans, Y. Bengio, and L. Bottou, editors, *Adv. in Neural Inform. Processing Syst. 21*, pages 897–904. 2009.

W. Li and A. McCallum. Semi-supervised sequence modeling with syntactic topic models. In *Proc. of the 20th Nat. Conf. on Artificial Intelligence*, volume 2, pages 813–818. AAAI Press, 2005.

L. Lovasz and S. Vempala. Fast algorithms for logconcave functions: Sampling, rounding, integration and optimization. In *Proc. of the 47th Ann. IEEE Symp. on Foundations of Comput. Sci.*, pages 57–68. IEEE Computer Society, 2006. ISBN 0-7695-2720-5.

P. Miettinen, T. Mielikäinen, A. Gionis, G. Das, and H. Mannila. The discrete basis problem. *IEEE Trans. Knowl. Data Eng.*, 20(10):1348–1362, 2008.

I. Mukherjee and D. M. Blei. Relative performance guarantees for approximate inference in latent Dirichlet allocation. In D. Koller, D. Schuurmans, Y. Bengio, and L. Bottou, editors, *Adv. in Neural Inform. Processing Syst. 21*, pages 1129–1136. 2009.

I. Porteous, D. Newman, A. Ihler, A. Asuncion, P. Smyth, and M. Welling. Fast collapsed gibbs sampling for latent dirichlet allocation. In *Proc. of the 14th ACM SIGKDD Int. Conf. on Knowledge Discovery and Data Mining*, pages 569–577, New York, NY, USA, 2008. ACM.

A. Schrijver. *Combinatorial optimization. Polyhedra and efficiency. Vol. A*, volume 24 of *Algorithms and Combinatorics*. Springer-Verlag, Berlin, 2003. ISBN 3-540-44389-4. Paths, flows, matchings, Chapters 1–38.

J. K. Seppänen, E. Bingham, and H. Mannila. A simple algorithm for topic identification in 0-1 data. In *Proc. of the 7th European Conf. on Principles and Practice of Knowledge Discovery in Databases*, pages 423–434. Springer-Verlag, 2003.

Y. W. Teh, D. Newman, and M. Welling. A collapsed variational Bayesian inference algorithm for latent Dirichlet allocation. In *Adv. in Neural Inform. Processing Syst. 19*, volume 19, 2007.

A. L. Yuille and A. Rangarajan. The concave-convex procedure. *Neural Comput.*, 15:915–936, April 2003. ISSN 0899-7667.

